# Structure Learning for Optimization

**Shulin (Lynn) Yang**
Department of Computer Science
University of Washington
Seattle, WA 98195
yang@cs.washington.edu

**Ali Rahimi**
Red Bow Labs
Berkeley, CA 94704
ali@redbowlabs.com

## Abstract

We describe a family of global optimization procedures that automatically decompose optimization problems into smaller loosely coupled problems. The solutions of these are subsequently combined with message passing algorithms. We show empirically that these methods produce better solutions with fewer function evaluations than existing global optimization methods. To develop these methods, we introduce a notion of coupling between variables of optimization. This notion of coupling generalizes the notion of independence between random variables in statistics, sparseness of the Hessian in nonlinear optimization, and the generalized distributive law. Despite its generality, this notion of coupling is easier to verify empirically, making structure estimation easy, while allowing us to migrate well-established inference methods on graphical models to the setting of global optimization.

## 1   Introduction

We consider optimization problems where the objective function is costly to evaluate and may be accessed only by evaluating it at requested points. In this setting, the function is a black box, and have no access to its derivative or its analytical structure. We propose solving such optimization problems by first estimating the internal structure of the black box function, then optimizing the function with message passing algorithms that take advantage of this structure. This lets us solve global optimization problems as a sequence of small grid searches that are coordinated by dynamic programming. We are motivated by the problem of tuning the parameters of computer programs to improve their accuracy or speed. For the programs that we consider, it can take several minutes to evaluate these performance measures under a particular parameter setting.

Many optimization problems exhibit only loose coupling between many of the variables of optimization. For example, to tune the parameters of an audio-video streaming program, the parameters of the audio codec could conceivably be tuned independently of the parameters of the video codec. Similarly, to tune the networking component that glues these codecs together it suffices to consider only a few parameters of the codecs, such as their output bit-rate. Such notions of conditional decoupling are conveniently depicted in a graphical form that represents the way the objective function factors into a sum or product of terms each involving only a small subset of the variables. This factorization structure can then be exploited by optimization procedures such as dynamic programming on trees or junction trees. Unfortunately, the factorization structure of a function is difficult to estimate from function evaluation queries only.

We introduce a notion of decoupling that can be more readily estimated from function evaluations. At the same time, this notion of decoupling is more general than the factorization notion of decoupling in that functions that do not factorize may still exhibit this type of decoupling. We say that two variables are decoupled if the optimal setting of one variable does not depend on the setting of the other. This is formalized below in a way that parallels the notion of conditional decoupling between random variables in statistics. This parallel allows us to migrate much of the machinery developed

for inference on graphical models to global optimization . For example, decoupling can be visualized with a graphical model whose semantics are similar to those of a Markov network. Analogs of the max-product algorithm on trees, the junction tree algorithm, and loopy belief propagation can be readily adapted to global optimization. We also introduce a simple procedure to estimate decoupling structure.

The resulting recipe for global optimization is to first estimate the decoupling structure of the objective function, then to optimize it with a message passing algorithm that utilises this structure. The message passing algorithm relies on a simple grid search to solve the sub-problems it generates. In many cases, using the same number of function evaluations, this procedure produces solutions with objective values that improve over those produced by existing global optimizers by as much as 10%. This happens because knowledge of the independence structure allows this procedure to explore the objective function only along directions that cause the function to vary, and because the grid search that solves the sub-problems does not get stuck in local minima.

## 2   Related work

The idea of estimating and exploiting loose coupling between variables of optimization appears implicitly in Quasi-Newton methods that numerically estimate the Hessian matrix, such as BFGS (Nocedal & Wright, 2006, Chap. 6). Indeed, the sparsity pattern of the Hessian indicates the pairs of terms that do not interact with each other in a second-order approximation of the function. This is strictly a less powerful notion of coupling than the factorization model, which we argue below, is in turn less powerful than our notion of decoupling.

Others have proposed approximating the objective function while simultaneously optimizing over it Srinivas et al. (2010). The procedure we develop here seeks only to approximate decoupling structure of the function, a much simpler task to carry out accurately.

A similar notion of decoupling has been explored in the decision theory literature Keeney & Raiffa (1976); Bacchus & Grove (1996), where decoupling was used to reason about preferences and utilities during decision making. In contrast, we use decoupling to solve black-box optimization problems and present a practical algorithm to estimate the decoupling structure.

## 3   Decoupling between variables of optimization

A common way to minimize an objective function over many variables is to factorize it into terms, each of which involves only a small subset of the variables Aji & McEliece (2000). Such a representation, if it exists, can be optimized via a sequence of small optimization problems with dynamic programming. This insight motivates message passing algorithms for inference on graphical models. For example, rather than minimizing the function $f_1(x, y, z) = g_1(x, y) + g_2(y, z)$ over its three variables simultaneously, one can compute the function $g_3(y) = \min_z g_2(y, z)$, then the function $g_4(x) = \min_y g_1(x, y) + g_3(y)$, and finally minimizing $g_4$ over $x$. A similar idea works for the function $f_2(x, y, z) = g_1(x, y) g_2(y, z)$ and indeed, whenever the operator that combines the factors is associative, commutative, and allows the "min" operator to distribute over it.

However, it is not necessary for a function to factorize for it to admit a simple dynamic programming procedure. For example, a factorization for the function $f_3(x, y, z) = x^2 y^2 z^2 + x^2 + y^2 + z^2$ is elusive, yet the arguments of $f_3$ are decoupled in the sense that the setting of any two variables does not affect the optimal setting of the third. For example, $\text{argmin}_x f_3(x, y_0, z_0)$ is always $x = 0$, and similarly for $y$ and $z$. This decoupling allows us to optimize over the variables separately. This is not a trivial property. For example, the function $f_4(x, y, z) = (x - y)^2 + (y - z)^2$ exhibits no such decoupling between $x$ and $y$ because the minimizer of $\text{argmin}_x f_4(x, y_0, z_0)$ is $y_0$, which is obviously a function of the second argument of $f$. The following definition formalizes this concept:

**Definition 1** (Blocking and decoupling). *Let $f : \Omega \to \mathbb{R}$ be a function on a compact domain and let $\mathcal{X} \times \mathcal{Y} \times \mathcal{Z} \subseteq \Omega$ be a subset of the domain. We say that the coordinates $\mathcal{Z}$ block $\mathcal{X}$ from $\mathcal{Y}$ under $f$ if the set of minimizers of $f$ over $\mathcal{X}$ does not change for any setting of the variables $\mathcal{Y}$ given a setting of the variables $\mathcal{Z}$:*

$$\forall_{\substack{Y_1 \in \mathcal{Y}, Y_2 \in \mathcal{Y} \\ Z \in \mathcal{Z}}} \underset{X \in \mathcal{X}}{\text{argmin}} \, f(X, Y_1, Z) = \underset{X \in \mathcal{X}}{\text{argmin}} \, f(X, Y_2, Z).$$

*We will say that $\mathcal{X}$ and $\mathcal{Y}$ are decoupled conditioned on $\mathcal{Z}$ under $f$, or $\mathcal{X} \perp_f \mathcal{Y} | \mathcal{Z}$, if $\mathcal{Z}$ blocks $\mathcal{X}$ from $\mathcal{Y}$ and $\mathcal{Z}$ blocks $\mathcal{Y}$ from $\mathcal{X}$ under $f$ at the same time.*

*We will simply say that $\mathcal{X}$ and $\mathcal{Y}$ are decoupled, or $\mathcal{X} \perp_f \mathcal{Y}$, when $\mathcal{X} \perp_f \mathcal{Y}\big|\mathcal{Z}$, $\Omega = \mathcal{X} \times \mathcal{Y} \times \mathcal{Z}$, and $f$ is understood from context.*

For a given function $f(x_1, \ldots, x_n)$, decoupling between the variables can be represented graphically with an undirected graph analogous to a Markov network:

**Definition 2.** *A graph $G = (\{x_1, \ldots, x_n\}, E)$ is a coupling graph for a function $f(x_1, \ldots, x_n)$ if $(i, j) \notin E$ implies $x_i$ and $x_j$ are decoupled under $f$.*

The following result mirrors the notion of separation in Markov networks and makes it easy to reason about decoupling between groups of variables with coupling graphs (see the appendix for a proof):

**Proposition 1.** *Let $\mathcal{X}, \mathcal{Y}, \mathcal{Z}$ be groups of nodes in a coupling graph for a function $f$. If every path from a node in $\mathcal{X}$ to a node in $\mathcal{Y}$ passes through a node in $\mathcal{Z}$, then $\mathcal{X} \perp_f \mathcal{Y}\big|\mathcal{Z}$.*

Functions that factorize as a product of terms exhibit this type of decoupling. For subsets of variables $\mathcal{X}, \mathcal{Y}, \mathcal{Z}$, we say $\mathcal{X}$ is conditionally separated from $\mathcal{Y}$ by $\mathcal{Z}$ by factorization, or $\mathcal{X} \perp_\otimes \mathcal{Y}\big|\mathcal{Z}$, if $\mathcal{X}$ and $\mathcal{Y}$ are separated in that way in the Markov network induced by the factorization of $f$. The following is a generalization of the familiar result that factorization implies the global Markov property (Koller & Friedman, 2009, Thm. 4.3) and follows from Aji & McEliece (2000):

**Theorem 1** (Factorization implies decoupling)**.** *Let $f(x_1, \ldots, x_n)$ be a function on a compact domain, and let $\mathcal{X}_1, \ldots, \mathcal{X}_S, \mathcal{X}, \mathcal{Y}, \mathcal{Z}$ be subsets of $\{x_1, \ldots, x_n\}$. Let $\otimes$ be any commutative associative semi-ring operator over which the min operator distributes. If $f$ factorizes as $f(x_1, \ldots, x_n) = \otimes_{s=1}^{S} g_s(X_s)$, then $\mathcal{X} \perp_f \mathcal{Y}\big|\mathcal{Z}$ whenever $\mathcal{X} \perp_\otimes \mathcal{Y}\big|\mathcal{Z}$.*

However decoupling is strictly more powerful than factorization. While $\mathcal{X} \perp_\otimes \mathcal{Y}$ implies $\mathcal{X} \perp_f \mathcal{Y}$, the reverse is not necessarily true: there exist functions that admit no factorization at all, yet whose arguments are completely mutually decoupled. Appendix B gives an example.

# 4 Optimization procedures that utilize decoupling

When a cost function factorizes, dynamic programming algorithms can be used to optimize over the variables Aji & McEliece (2000). When a cost function exhibits decoupling as defined above, the same dynamic programming algorithms can be applied with a few minor modifications.

The algorithms below refer to a function $f$ whose arguments are partitioned over the sets $\mathcal{X}_1, \ldots, \mathcal{X}_n$. Let $X_i^*$ denote the optimal value of $X_i \in \mathcal{X}_i$. We will take simplifying liberties with the order of the arguments of $f$ when this causes no ambiguity. We will also replace the variables that do not participate in the optimization (per decoupling) with an ellipsis.

## 4.1 Optimization over trees

Suppose the coupling graph between some partitioning $\mathcal{X}_1, \ldots, \mathcal{X}_m$ of the arguments of $f$ is tree-structured, in the sense that $\mathcal{X}_i \perp_f \mathcal{X}_j$ unless the edge $(i, j)$ is in the tree. To optimize over $f$ with dynamic programming, define $\mathcal{X}_0$ arbitrarily as the root of the tree, let $p_i$ denote the index of the parent of $\mathcal{X}_i$, and let $C_i^1, C_i^2, \ldots$ denote the indices of its children. At each leaf node $\ell$, construct the functions

$$\hat{X}_\ell(X_{p_\ell}) := \underset{X_\ell \in \mathcal{X}_\ell}{\operatorname{argmin}} f(X_\ell, X_{p_\ell}). \tag{1}$$

By decoupling, the optimal value of $X_\ell$ depends only on the optimal value of its parent, so $X_\ell^* = \hat{X}_\ell(X_{p_\ell}^*)$.

For all other nodes $i$, define recursively starting from the parents of the leaf nodes the functions

$$\hat{X}_i(X_{p_i}) = \underset{X_i \in \mathcal{X}_i}{\operatorname{argmin}} f(X_i, X_{p_i}, \hat{X}_{C_i^1}(X_i), \hat{X}_{C_i^2}(X_i), \ldots) \tag{2}$$

Again, the optimal value of $X_i$ depends only on the optimal setting of its parent, $X_{p_i}^*$, and it can be verified that $X_i^* = \hat{X}_i(X_{p_i}^*)$.

In our implementation of this algorithm, to represent a function $\hat{X}_i(X)$, we discretize its argument into a grid, and store the function as a table. To compute the entries of the table, a subordinate global optimizer computes the minimization that appears in the definition of $\hat{X}_i$.

## 4.2 Optimization over junction trees

Even when the coupling graph for a function is not tree-structured, a thin junction tree can often be constructed for it. A variant of the above algorithm that mirrors the junction tree algorithm can be used to efficiently search for the optima of the function.

Recall that a tree $T$ of cliques is a junction tree for a graph $G$ if it satisfies the following three properties: there is one path between each pair of cliques; for each clique $C$ of $G$ there is some clique $A$ in $T$ such that $C \subseteq A$; for each pair of cliques $A$ and $B$ in $T$ that contain node $i$ of $G$, each clique on the unique path between $A$ and $B$ also contains $i$.

These properties guarantee that $T$ is tree-structured, that it covers all nodes and edges in $G$, and that two nodes $v$ and $u$ in two different cliques $\mathcal{X}_i$ and $\mathcal{X}_j$ are decoupled from each other conditioned on the union of the cliques on the path between $u$ and $v$ in $T$. Many heuristics exist for constructing a thin junction tree for a graph Jensen & Graven-Nielsen (2007); Huang & Darwiche (1996).

To search for the minimizers of $f$, using a junction tree for its coupling graph, denote by $\mathcal{X}_{ij} := \mathcal{X}_i \cap \mathcal{X}_j$ the intersection of the groups of variables $\mathcal{X}_i$ and $\mathcal{X}_j$ and by $\mathcal{X}_{i \setminus j} = \mathcal{X}_i \setminus \mathcal{X}_j$ the set of nodes in $\mathcal{X}_i$ but not in $\mathcal{X}_j$. At every leaf clique $\ell$ of the junction tree, construct the function

$$\hat{X}_\ell (X_{\ell,p_\ell}) := \operatorname*{argmin}_{X_{\ell \setminus p_\ell} \in \mathcal{X}_{\ell \setminus p_\ell}} f(X_\ell). \tag{3}$$

For all other cliques $i$, compute recursively starting from the parents of the leaf cliques

$$\hat{X}_i(X_{i,p_i}) = \operatorname*{argmin}_{X_{i,p_i} \in \mathcal{X}_{i \setminus p_i}} f(X_i, \hat{X}_{C_i^1}(X_{i,C_i^1}), \hat{X}_{C_i^2}(X_{i,C_i^2}), \ldots). \tag{4}$$

As before, decoupling between the cliques, conditioned on the intersection of the cliques, guarantees that $\hat{X}_i(X_{i,p_i}^*) = X_i^*$. And as before, our implementation of this algorithm stores the intermediate functions as tables by discretizing their arguments.

## 4.3 Other strategies

When the cliques of the junction tree are large, the subordinate optimizations in the above algorithm become costly. In such cases, the following adaptations of approximate inference algorithms are useful:

- The algorithm of Section 4.1 can be applied to a maximal spanning tree of the coupling graph.
- Analogously to Loopy Belief Propagation Pearl (1997), an arbitrary neighbor of each node can be declared as its parent, and the steps of Section 4.1 can be applied to each node until convergence.
- Loops in the coupling graph can be broken by conditioning on a node in each loop, resulting in a tree-structured coupling graph conditioned on those nodes. The optimizer of Section 4.1 then searches for the minima conditioned on the value of those nodes in the inner loop of a global optimizer that searches for good settings for the conditioned nodes.

## 5 Graph structure learning

It is possible to estimate decoupling structure between the arguments of a function $f$ with the help of a subordinate optimizer that only evaluates $f$.

A straightforward application of definition 1 to assess empirically whether groups of variables $\mathcal{X}$ and $\mathcal{Y}$ are decoupled conditioned on a group of variables $\mathcal{Z}$ would require comparing the minimizer of $f$ over $\mathcal{X}$ for every possible value of $\mathcal{Z}$ and $\mathcal{Y}$. This is not practical because it is at least as difficult as minimizing $f$. Instead, we rely on the following proposition, which follows directly from 1:

**Proposition 2** (Invalidating decoupling). *If for some $Z \in \mathcal{Z}$ and $Y_0, Y_1 \in \mathcal{Y}$, we have $argmin_{X \in \mathcal{X}} f(X, Y_0, Z) \neq argmin_{X \in \mathcal{X}} f(X, Y_1, Z)$, then $\mathcal{X} \not\perp_f \mathcal{Y} | \mathcal{Z}$.*

Following this result, an approximate coupling graph can be constructed by positing and invalidating decoupling relations. Starting with a graph containing no edges, we consider all groupings $\mathcal{X} =$

$\{x_i\}, \mathcal{Y} = \{x_j\}, \mathcal{Z} = \Omega \setminus \{x_i, x_j\}$, of variables $x_1, \ldots, x_n$. We posit various values of $Z \in \mathcal{Z}, Y_0 \in \mathcal{Y}$ and $Y_1 \in \mathcal{Y}$ under this grouping, and compute the minimizers over $X \in \mathcal{X}$ of $f(X, Y_0, Z)$ and $f(X, Y_1, Z)$ with a subordinate optimizer. If the minimizers differ, then by the above proposition, $\mathcal{X}$ and $\mathcal{Y}$ are not decoupled conditioned on $\mathcal{Z}$, and an edge is added between $x_i$ and $x_j$ in the graph. Algorithm 1 summarizes this procedure.

---

**Algorithm 1** Estimating the coupling graph of a function.

---

**input** A function $f : \mathcal{X}_1 \times \cdots \mathcal{X}_n \to \mathbb{R}$, with $\mathcal{X}_i$ compact; A discretization $\hat{\mathcal{X}}_i$ of $\mathcal{X}_i$; A similarity threshold $\epsilon > 0$; The number of times, $N_Z$, to sample $Z$.
**output** A coupling graph $G = ([x_1, \ldots, x_n], E)$.
   $E \leftarrow \emptyset$
   **for** $i, j \in [1, \ldots, n]$; $y_0, y_1 \in \hat{\mathcal{X}}_j$; $1 \ldots N_Z$ **do**
      $Z \sim U(\hat{\mathcal{X}}_1 \times \cdots \times \hat{\mathcal{X}}_n \setminus \hat{\mathcal{X}}_i \times \hat{\mathcal{X}}_j)$
      $\hat{x}_0 \leftarrow \operatorname{argmin}_{x \in \hat{\mathcal{X}}_i} f(x, y_0, Z)$; $\hat{x}_1 \leftarrow \operatorname{argmin}_{x \in \hat{\mathcal{X}}_i} f(x, y_1, Z)$
      **if** $\|\hat{x}_0 - \hat{x}_1\| \geq \epsilon$ **then**
         $E \leftarrow E \cup \{(i, j)\}$
      **end if**
   **end for**

---

In practice, we find that decoupling relationships are correctly recovered if values of $Y_0$ and $Y_1$ are chosen by quantizing $\mathcal{Y}$ into a set $\hat{\mathcal{Y}}$ of 4 to 10 uniformly spaced discrete values and exhaustively examining the settings of $Y_0$ and $Y_1$ in $\hat{\mathcal{Y}}$. A few values of $Z$ (fewer than five) sampled uniformly at random from a similarly discretized set $\hat{\mathcal{Z}}$ suffice.

## 6  Experiments

We evaluate a two step process for global optimization: first estimating decoupling between variables using the algorithm of Section 5, then optimizing with this structure using an algorithm from Section 4. Whenever Algorithm 1 detects tree-structured decoupling, we use the tree optimizer of Section 4.1. Otherwise we either construct a junction tree and apply the junction tree optimizer of Section 4.2 if the junction tree is thin, or we approximate the graph with a maximum spanning tree and apply the tree solver of Section 4.1.

We compare this approach with three state-of-the-art black-box optimization procedures: Direct Search Perttunen et al. (1993) (a deterministic space carving strategy), FIPS Mendes et al. (2004) (a biologically inspired randomized algorithm), and MEGA Hazen & Gupta (2009) (a multiresolution search strategy with numerically computed gradients). We use a publicly available implementation of Direct Search [1], and an implementation of FIPS and MEGA available from the authors of MEGA. We set the number of particles for FIPS and MEGA to the square of the dimension of the problem plus one, following the recommendation of their authors.

As the subordinate optimizer for Algorithm 1, we use a simple grid search for all our experiments. As the subordinate optimizer for the algorithms of Section 4, we experiment with grid search and the aforementioned state-of-the-art global optimizers.

We report results on both synthetic and real optimization problems. For each experiment, we report the quality of the solution each algorithm produces after a preset number of function calls. To vary the number of function calls the baseline methods invoke, we vary the number of time they iterate. Since our method does not iterate, we vary the number of function calls its subordinate optimizer invokes (when the subordinate optimizer is grid search, we vary the grid resolution).

The experiments demonstrate that using grid search as a subordinate strategy is sufficient to produce better solutions than all the other global optimizers we evaluated.

Table 1: Value of the iterates of the functions of Table 2 after 10,000 function evaluations (for our approach, this includes the function evaluations for structure learning). MIN is the ground truth optimal value when available. GR is the number of discrete values along each dimension for optimization. Direct Search (DIR), FIPS and MEGA are three state-of-the-art algorithms for global optimization.

| Function (n=50) | min | GR | Ours | DIR | FIPS | MEGA |
|---|---|---|---|---|---|---|
| Colville | 0 | 100 | **0** | $3e$-6 | $2e$-14 | 3.75 |
| Levy | 0 | 400 | **0.013** | 2.80 | 4.20 | 3.22 |
| Michalewics | n/a | 400 | **-48.9** | -18.2 | -18.4 | $-1.3e$-3 |
| Rastrigin | 0 | 400 | **0** | **0** | 23.6 | $4.2e$-3 |
| Schwefel | 0 | 400 | **8.6** | $1.9e4$ | $1.6e4$ | $1.4e4$ |
| Dixon&Price | 0 | 20 | 1 | **0.667** | 16.8 | 0.914 |
| Rosenbrock | 0 | 20 | **0** | $2.9e4$ | $5.7e4$ | 48.4 |
| Trid | n/a | 20 | **-2.2e4** | -185 | $3.3e4$ | -41 |
| Powell | 0 | 6 | 19.4 | 324 | 121 | **0.014** |

## 6.1 Synthetic objective functions

We evaluated the above strategies on a standard benchmark of synthetic optimization problems [2] shown in Appendix A. These are functions of 50 variables and are used as black-box functions in our experiments. In these experiments, the subordinate grid search of Algorithm 1 discretized each dimension into four discrete values. The algorithms of Section 4 also used grid search as a subordinate optimizer. For this grid search, each dimension was discretized into $GR = \left( \frac{E_{max}}{N_{mc}} \right)^{\frac{1}{S_{mc}}}$ discrete values where $E_{max}$ is a cap on the number of function evaluations to perform, $S_{mc}$ is the size of the largest clique in the junction tree, and $N_{mc}$ is the number of nodes in the junction tree.

Figure 1 shows that in all cases, Algorithm 1 recovered decoupling structure exactly even for very coarse grids. Values of $N_Z$ greater than 1 did not improve the quality of the recovered graph, justifying our heuristic of keeping $N_Z$ small. We used $N_Z = 1$ in the remainder of this subsection.

Table 1 summarizes the quality of the solutions produced by the various algorithms after 10,000 function evaluations. Our approach outperformed the others on most of these problems. As expected, it performed particularly well on functions that exhibit sparse coupling, such as Levy, Rastrigin, and Schwefel.

In addition to achieving better solutions given the same number of function evaluations, our approach also imposed lower computational overhead than the other methods: to process the entire benchmark of this section takes our approach 2.2 seconds, while Direct Search, FIPS and MEGA take 5.7 minutes, 3.7 minutes and 53.3 minutes respectively.

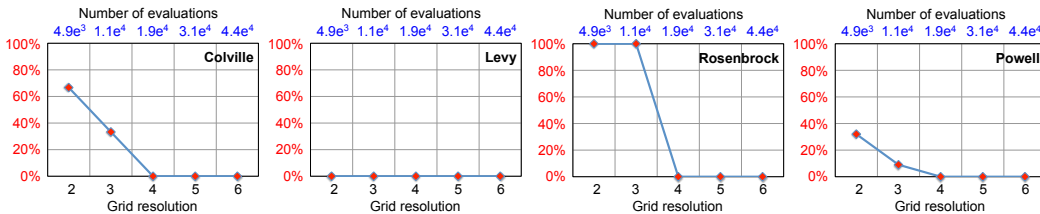

Figure 1: Very coarse gridding is sufficient in Algorithm 1 to correctly recover decoupling structure. The plots show percentage of incorrectly recovered edges in the coupling graph on four synthetic cost functions as a function of the grid resolution (bottom x-axis) and the number of function evaluations (top x-axis). $N_Z = 1$ in these experiments.

## 6.2 Experiments on real applications

We considered the real-world problem of automatically tuning the parameters of machine vision and machine learning programs to improve their accuracy on new datasets. We sought to tune the

parameters of a face detector, a document topic classifier, and a scene recognizer to improve their accuracy on new application domains. Automatic parameter tuning allows a user to quickly tune a program's default parameters to their specific application domain without tedious trial and error. To perform this tuning automatically, we treated the accuracy of a program as a black box function of the parameter values passed to it. These were challenging optimization problems because the derivative of the function is elusive and each function evaluation can take minutes. Because the output of a program tends to depend in a structured way on its parameters, our method achieved significant speedups over existing global optimizers.

### 6.2.1 Face detection

The first application was a face detector. The program has five parameters: the size, in pixels, of the smallest face to consider, the minimum distance, in pixels, between detected faces; a floating point subsampling rate for building a multiresolution pyramid of the input image; a boolean flag that determines whether to apply non-maximal suppression; and the choice of one of four wavelets to use. Our goal was to minimize the detection error rate of this program on the *GENKI-SZSL* dataset of 3,500 faces [3]. Depending on the parameter settings, evaluating the accuracy of the program on this dataset takes between 2 seconds and 2 minutes.

Algorithm 1 was run with a grid search as a subordinate optimizer with three discrete values along the continuous dimensions. It invoked 90 function evaluations and produced a coupling graph wherein the first three of the above parameters formed a clique and where the remaining two parameter were decoupled of the others. Given this coupling graph, our junction tree optimizer with grid search (with the continuous dimensions quantized into 10 discrete values) invoked 1000 function evaluations, and found parameter settings for which the accuracy of the detector was 7% better than the parameter settings found by FIPS and Direct Search after the same number of function evaluations. FIPS and Direct Search fail to improve their solution even after 1800 evaluations. MEGA fails to improve over the initial detection error of 50.84% with any number of iterations. To evaluate the accuracy of our method under different numbers of function invocations, we varied the grid resolution between 2 to 12. See Figure 2. These experiments demonstrate how a grid search can help overcome local minima that cause FIPS and Direct Search to get stuck.

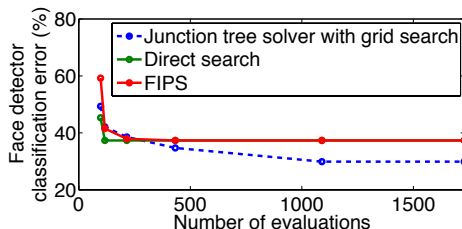

Figure 2: Depending on the number of function evaluations allowed, our method produces parameter settings for the face detector that are better than those recovered by FIPS or Direct Search by as much as 7%.

### 6.2.2 Scene recognition

The second application was a visual scene recognizer. It extracts GIST features Oliva & Torralba (2001) from an input image and classifies these features with a linear SVM. Our task was to tune the six parameters of GIST to improve the recognition accuracy on a subset of the *LabelMe* dataset [4], which includes images of scenes such as coasts, mountains, streets, etc. The parameters of the recognizer include a radial cut-off frequency (in cycles/pixel) of a circular filter that reduces illumination effects, the number of bins in a radial histogram of the response of a spatial spacial filter, and the number of image regions in which to compute these histograms. Evaluating the classification error under a set of parameters requires extracting GIST features with these parameters on a training set, training a linear SVM, then applying the extractor and classifier to a test set. Each evaluation takes between 10 and 20 minutes depending on the parameter settings.

Algorithm 1 was run with a grid search as the subordinate optimizer, discretizing the search space into four discrete values along each dimension. This results in a graph that admits no thin junction tree, so we approximate it with a maximal spanning tree. We then apply the tree optimizer of Section 4.1 using as subordinate optimizers Direct Search, FIPS, and grid search (with five discrete values along each dimension). After a total of roughly 300 function evaluations, the tree optimizer with FIPS produces parameters that result in a classification error of 29.17%. With the same number of function evaluations, *Direct Search* and *FIPS* produce parameters that resulted in classification errors of 33.33% and 31.13% respectively. The tree optimizer with Direct Search and grid search as subordinate optimizers resulted in error rates of 31.72% and 33.33%.

In this application, the proposed method enjoys only modest gains of $\sim 2\%$ because the variables are tightly coupled, as indicated by the denseness of the graph and the thickness of the junction tree.

### 6.2.3  Multi-class classification

The third application was to tune the hyperparameters of a multi-class SVM classifier on the *RCV1-v2* text categorization dataset [5]. This dataset consists of a training set of 23,149 documents and a test set of 781,265 documents each labeled with one of 101 topics Lewis et al. (2004). Our task was to tune the 101 regularization parameters of the 1 vs. all classifiers that comprise a multi-class classifier. The objective was the so-called macro-average $F$-score Tague (1981) on the test set. The $F$ score for one category is $F = 2rp/(r + p)$, where $r$ and $p$ are the recall and precision rates for that category. The macro-average $F$ score is the average of the $F$ scores over all categories. Each evaluation requires training the classifier using the given hyperparameters and evaluating the resulting classifier on the test set, and takes only a second since the text features have been pre-computed.

Algorithm 1 with grid search as a subordinate optimizer with a grid resolution of three discrete values along each dimension found no coupling between the hyperparameters. As a result, the algorithms of Section 4.1 reduce to optimizing over each one-dimensional parameter independently. We carried out these one-dimensional optimizations with Direct Search, FIPS, and grid search (discretizing each dimension into 100 values). After roughly 100,000 evaluations, these resulted in similar scores of $F = 0.6764, 0.6720$, and $0.6743$, respectively. But with the same number of evaluations, off-the-shelf Direct Search and FIPS result in scores of $F = 0.6324$ and $0.6043$, respectively, nearly 11% worse.

The cost of estimating the structure in this problem was large, since it grows quadratically with the number of classes, but worth the effort because it indicated that each variable should be optimized independently, ultimately resulting in huge speedups [6].

## 7  Conclusion

We quantified the coupling between variables of optimization in a way that parallels the notion of independence in statistics. This lets us identify decoupling between variables in cases where the function does not factorize, making it strictly stronger than the notion of decoupling in statistical estimation. This type of decoupling is also easier to evaluate empirically. Despite these differences, this notion of decoupling allows us to migrate to global optimization many of the message passing algorithms that were developed to leverage factorization in statistics and optimization. These include belief propagation and the junction tree algorithm. We show empirically that optimizing cost functions by applying these algorithms to an empirically estimated decoupling structure outperforms existing black box optimization procedures that rely on numerical gradients, deterministic space carving, or biologically inspired searches. Notably, we observe that it is advantageous to decompose optimization problems into a sequence of small deterministic grid searches using this technique, as opposed to employing existing black box optimizers directly.

## Footnotes

[1]Available from `http://www4.ncsu.edu/~ctk/Finkel_Direct/`.

[2]Acquired from `http://www-optima.amp.i.kyoto-u.ac.jp/member/student/hedar/Hedar_files/go.htm`.

[3]Available from `http://mplab.ucsd.edu`.

[4]Available from `http://labelme.csail.mit.edu`.

[5] Available from `http://trec.nist.gov/data/reuters/reuters.html`.

[6] After running these experiments, we discovered a result of Fan & Lin (2007) showing that optimizing the macro-average F-measure is equivalent to optimizing per-category F-measure, thereby validating decoupling structure recovered by Algorithm 1.

# References

Aji, S. and McEliece, R. The generalized distributive law and free energy minimization. *IEEE Transaction on Informaion Theory*, 46(2), March 2000.

Bacchus, F. and Grove, A. Utility independence in a qualitative decision theory. *In Proceedings of the 6th International Conference on Principles of Knowledge Representation and Reasoning*, pp. 542–552, 1996.

Fan, R. E. and Lin, C. J. A study on threshold selection for multi-label classification. Technical report, National Taiwan University, 2007.

Hazen, M. and Gupta, M. Gradient estimation in global optimization algorithms. *Congress on Evolutionary Computation*, pp. 1841–1848, 2009.

Huang, C. and Darwiche, A. Inference in belief networks: A procedural guide. *International Journal of Approximate Reasoning*, 15(3):225–263, 1996.

Jensen, F. and Graven-Nielsen, T. *Bayesian Networks and Decision Graphs*. Springer, 2007.

Keeney, R. L. and Raiffa, H. *Decisions with Multiple Objectives: Preferences and Value Trade-offs*. Wiley, 1976.

Koller, D. and Friedman, N. *Probabilistic Graphical Models: Principles and Techniques*. MIT Press, 2009.

Lewis, D., Yang, Y., Rose, T., and Li, F. RCV1: A new benchmark collection for text categorization research. *Journal of Machine Learning Research*, 2004.

Mendes, R., Kennedy, J., and Neves, J. The fully informed particle swarm: Simpler, maybe better. *IEEE Transactions on Evolutionary Computation*, 1(1):204–210, 2004.

Nocedal, J. and Wright, S. *Numerical Optimization*. Springer, 2nd edition, 2006.

Oliva, A. and Torralba, A. Modeling the shape of the scene: a holistic representation of the spatial envelope. *International Journal of Computer Vision*, 43:145–175, 2001.

Pearl, J. *Probabilistic Reasoning in Intelligent Systems: Networks of Plausible Inference*. Morgan Kaufmann, 1997.

Perttunen, C., Jones, D., and Stuckman, B. Lipschitzian optimization without the Lipschitz constant. *Journal of Optimization Theory and Application*, 79(1):157–181, 1993.

Srinivas, N., Krause, A., Kakade, S., and Seeger, M. Gaussian process optimization in the bandit setting: No regret and experimental design. In *International Conference on Machine Learning (ICML)*, 2010.

Tague, J. M. The pragmatics of information retrieval experimentation. *Information Retrieval Experiment*, pp. 59–102, 1981.

